# Sequential Tracking in Pricing Financial Options using Model Based and Neural Network Approaches

**Mahesan Niranjan**
Cambridge University Engineering Department
Cambridge CB2 1PZ, England
niranjan@eng.cam.ac.uk

## Abstract

This paper shows how the prices of option contracts traded in financial markets can be tracked sequentially by means of the Extended Kalman Filter algorithm. I consider call and put option pairs with identical strike price and time of maturity as a two output nonlinear system. The Black-Scholes approach popular in Finance literature and the Radial Basis Functions neural network are used in modelling the nonlinear system generating these observations. I show how both these systems may be identified recursively using the EKF algorithm. I present results of simulations on some FTSE 100 Index options data and discuss the implications of viewing the pricing problem in this sequential manner.

## 1  INTRODUCTION

Data from the financial markets has recently been of much interest to the neural computing community. The complexity of the underlying macro-economic system and how traders react to the flow of information leads to highly nonlinear relationships between observations. Further, the underlying system is essentially time varying, making any analysis both difficult and interesting. A number of problems, including forecasting a univariate time series from past observations, rating credit risk, optimal selection of portfolio components and pricing options have been thrown at neural networks recently.

The problem addressed in this paper is that of sequential estimation, applied to pricing of options contracts. In a nonstationary environment, such as financial markets, sequential estimation is the natural approach to modelling. This is because data arrives at the modeller sequentially, and there is the need to build and apply the

best possible model with available data. At the next point in time, some additional data is available and the task becomes one of optimally updating the model to account for the new data. This can either be done by reestimating the model with a moving window of data or by sequentially propagating the estimates of model parameters and some associated information (such as the error covariance matrices in the Kalman filtering framework discussed in this paper).

## 2 SEQUENTIAL ESTIMATION

Sequential estimation of nonlinear models via the Extended Kalman Filter algorithm is well known (*e.g.* Candy, 1986; Bar-Shalom & Li, 1993). This approach has also been widely applied to the training of Neural Network architectures (*e.g.* Kadirkamanathan & Niranjan, 1993; Puskorius & Feldkamp, 1994). In this section, I give the necessary equations for a second order EKF, *i.e.* Taylor series expansion of the nonlinear output equations, truncated at order two, for the state space model simplified to the system identification framework considered here.

The parameter vector or state vector, $\theta$, is assumed to have the following simple random walk dynamics.

$$\underline{\theta}(n+1) \ = \ \underline{\theta}(n) \ + \ \underline{u}(n),$$

where $\underline{u}(n)$ is a noise term, known as process noise. $\underline{u}(n)$ is of the same dimensionality as the number of states used to represent the system. The process noise gives a random walk freedom to the state dynamics facilitating the tracking behaviour desired in nonstationary environments. In using the Kalman filtering framework, we assume the covariance matrix of this noise process, denoted $Q$, is known. In practice, we set $Q$ to some small diagonal matrix.

The observations from the system are given by the equation

$$\underline{z}(n) \ = \ \underline{f}(\theta, \ \underline{U}) \ + \ \underline{w}(n),$$

where, the vector $\underline{z}(n)$ is the output of the system consisting of the call and put option prices at time $n$. $\underline{U}$ denotes the input information. In the problem considered here, $\underline{U}$ consists of the price of the underlying asset and the time to maturity if the option. $\underline{w}$ is known as the measurement noise, covariance matrix of which, denoted $R$, is also assumed to be known. Setting the parameters $R$ and $Q$ is done by trial and error and knowledge about the noise processes. In the estimation framework considered here, $Q$ and $R$ determine the tracking behaviour of the system. For the experiments reported in this paper, I have set these by trial and error, but more systematic approaches involving multiple models is possible (Niranjan *et al*, 1994).

The prior estimates at time $(n+1)$, using all the data upto time $(n)$ and the model of the dynamical system, or the prediction phase of the Kalman algorithm is given by the equations:

$$\hat{\underline{\theta}}(n+1|n) \ = \ \hat{\underline{\theta}}(n|n)$$
$$\hat{P}(n+1|n) \ = \ \hat{P}(n|n) \ + \ Q(n)$$
$$\hat{\underline{z}}(n+1|n) \ = \ \underline{J}_\theta(\underline{\theta}(n+1|n)) \ + \ \frac{1}{2}\sum_{i=1}^{n_\theta} \underline{e}_i \, \text{tr}\left(\underline{H}_{\theta\theta}^i(n+1) \, \hat{P}(n+1|n)\right)$$

where $\underline{J}_\theta$ and $\underline{H}_{\theta\theta}^i$ are the Jacobian and Hessians of the output $\underline{z}$; also $n_\theta = 2$. $\underline{e}_i$ are unit vectors in direction $i$. $\text{tr}(.)$ denotes trace of a matrix. The posterior esti-

mates or the correction phase of the Kalman algorithm are given by the equations:

$$
\begin{aligned}
S(n+1) &= \underline{J}_\theta(n+1)\hat{P}(n+1|n)\underline{J}'_\theta(n+1) \\
&\quad + \frac{1}{2}\sum_{i=1}^{n_\theta}\sum_{j=1}^{n_\theta}\underline{e}_i\underline{e}_j\left(\underline{H}^i_{\theta\theta}(n+1)\ \hat{P}(n+1|n)\underline{H}^j_{\theta\theta}(n+1)\ \hat{P}(n+1|n)\right) \\
&\quad + R \\
K(n+1) &= \hat{P}(n+1|n)\underline{J}_\theta(n+1)S^{-1}(n+1) \\
\underline{v}(n+1) &= \underline{z}(n+1)\ -\ \underline{J}_\theta(n+1)\hat{\underline{\theta}}(n+1|n) \\
\hat{\underline{\theta}}(n+1|n+1) &= \hat{\underline{\theta}}(n+1|n)\ +\ K(n+1)\underline{v}(n+1) \\
\hat{P}(n+1|n+1) &= (I\ -\ K(n+1)\underline{J}_\theta(n+1))\ \hat{P}(n+1|n)\ (I\ -\ K(n+1)\underline{J}_\theta(n+1))' \\
&\quad + K(n+1)\ R\ K(n+1)'
\end{aligned}
$$

Here, $K(n+1)$ is the Kalman Gain matrix and $\underline{v}(n+1)$ is the innovation signal.

## 3  BLACK-SCHOLES MODEL

The Black-Scholes equation for calculating the price of an European style call option (Hull, 1993) is

$$
C = S\,\mathcal{N}(d_1)\ -\ X\,e^{-r\,t_m}\,\mathcal{N}(d_2),
$$

where,

$$
\begin{aligned}
d_1 &= \frac{\ln(S/X)\ +\ (r\ +\ \frac{\sigma^2}{2})\sqrt{t_m}}{\sigma\sqrt{t_m}} \\
d_2 &= d_1\ -\ \sigma\sqrt{t_m}
\end{aligned}
$$

Here, $C$ is the price of the call option, $S$ the underlying asset price, $X$ the strike price of the option at maturity, $t_m$ the time to maturity and $r$ is the risk free interest rate. $\sigma$ is a term known as volatility and may be seen as an instantaneous variance of the time variation of the asset price. $\mathcal{N}(.)$ is the cumulative normal function. For a derivation of the formula and the assumptions upon which it is based see Hull, 1993. Readers unfamiliar with financial terms only need to know that all the quantities in the above equation, except $\sigma$, can be directly observed. $\sigma$ is usually estimated from a small moving window of data of about 50 trading days.

The equivalent formula for the price of a put option is given by

$$
P = -S\,\mathcal{N}(-d_1)\ +\ X\,e^{-r\,t_m}\,\mathcal{N}(-d_2),
$$

For recursive estimation of the option prices with this model, I assume that the instantaneous picture given by the Black Scholes model is correct. The state vector is two dimensional and consists of the volatility $\sigma$ and the interest rate $r$. The Jacobian and Hessian required for applying EKF algorithm are

$$
\underline{J}_\theta = \begin{pmatrix} \frac{\partial C}{\partial \sigma} & \frac{\partial C}{\partial r} \\[2mm] \frac{\partial P}{\partial \sigma} & \frac{\partial P}{\partial r} \end{pmatrix};\quad \mathrm{H}^1_{\theta\theta} = \begin{pmatrix} \frac{\partial^2 C}{\partial \sigma^2} & \frac{\partial^2 C}{\partial \sigma \partial r} \\[2mm] \frac{\partial^2 C}{\partial \sigma \partial r} & \frac{\partial^2 C}{\partial r^2} \end{pmatrix};\quad \mathrm{H}^2_{\theta\theta} = \begin{pmatrix} \frac{\partial^2 P}{\partial \sigma^2} & \frac{\partial^2 P}{\partial \sigma \partial r} \\[2mm] \frac{\partial^2 P}{\partial \sigma \partial r} & \frac{\partial^2 P}{\partial r^2} \end{pmatrix}
$$

Expressions for the terms in these matrices are given in table 1.

Table 1: First and Second Derivatives of the Black Scholes Model

| | |
|---|---|
| $\frac{\partial C}{\partial \sigma} = \frac{\partial P}{\partial \sigma}$ | $S \sqrt{t_m}\, \mathcal{N}'(d_1)$ |
| $\frac{\partial C}{\partial r}$ | $X t_m \exp(-r t_m)\mathcal{N}(d_2)$ |
| $\frac{\partial P}{\partial r}$ | $-X t_m \exp(-r t_m)\mathcal{N}(-d_2)$ |
| $\frac{\partial^2 C}{\partial \sigma^2} = \frac{\partial^2 P}{\partial \sigma^2}$ | $\frac{S\sqrt{t_m}d_1 d_2}{\sigma}\mathcal{N}'(d_1)$ |
| $\frac{\partial^2 C}{\partial r^2}$ | $-X t_m \exp(-r t_m)\left(t_m \mathcal{N}(d_2) - \frac{d_2\sqrt{t_m}}{\sigma}\mathcal{N}'(d_2)\right)$ |
| $\frac{\partial^2 P}{\partial r^2}$ | $X t_m \exp(-r t_m)\left(t_m \mathcal{N}(-d_2) - \frac{d_2\sqrt{t_m}}{\sigma}\mathcal{N}'(-d_2)\right)$ |
| $\frac{\partial^2 C}{\partial \sigma \partial r} = \frac{\partial^2 P}{\partial \sigma \partial r}$ | $\frac{-S d_1 t_m}{\sigma}\mathcal{N}'(d_1)$ |

## 4 NEURAL NETWORK MODELS

The data driven neural network model considered here is the Radial Basis Functions Network (RBF). Following Hutchinson *et al*, I use the following architecture:

$$\hat{C}/X = \sum_{j=1}^{m} \lambda_j\, \phi\left(\left(\underline{U} - \underline{\mu}_j\right)^t \Sigma^{-1} \left(\underline{U} - \underline{\mu}_j\right)\right) + \underline{W}^t \underline{Y} + w_0$$

where $\underline{U}$ is the two dimensional input data vector consisting of the asset price and time to maturity. The asset price $S$ is normalised by the strike price of the option $X$. The time to maturity, $t_m$, is also normalised such that the full lifetime of the option gets a value 1.0. These normalisations is the reason for considering options in pairs with the same strike price and time of maturity in this study. The nonlinear function $\phi(.)$ is set to $\phi(\alpha) = \sqrt{\alpha}$ and $m = 4$. With the nonlinear part of the network fixed, Kalman filter training of the RBF model is straightforward (see Kadirkamanathan & Niranjan, 1993). In the simulations studied in this paper, I used two approaches to fix the nonlinear functions. The first was to use the $\underline{\mu}_j$s and the $\Sigma$ published in Hutchinson *et al*. The second was to select the $\underline{\mu}_j$ terms as random subsets of the training data and set $\Sigma$ to $I$. The estimation problem is now linear and hence the Kalman filter equations become much simpler than the EKF equations used in the training of the Black-Scholes model.

In addition to training by EKF, I also implemented a batch training of the RBF model in which a moving window of data was used, training on data from $(n - 50)$ to $n$ days and testing on day $(n + 1)$. Since it is natural to assume that data closer to the test day is more appropriate than data far back in time, I incorporated a weighting function to weight the errors linearly, in the minimisation process. The least squares solution, with a weighting function, is given by the modified pseudo

Table 2: Comparison of the Approximation Errors for Different Methods

| Strike Price | Trivial | RBF Batch | RBF Kalman | BS Historic | BS Kalman |
|---|---|---|---|---|---|
| 2925 | 0.0790 | 0.0632 | 0.0173 | 0.0845 | 0.0180 |
| 3025 | 0.0999 | 0.1109 | 0.0519 | 0.1628 | 0.0440 |
| 3125 | 0.0764 | 0.0455 | 0.0193 | 0.0343 | 0.0112 |
| 3225 | 0.1116 | 0.0819 | 0.0595 | 0.0885 | 0.0349 |

inverse

$$\underline{l} = (Y' W Y)^{-1} Y' W \underline{t}$$

Matrix $W$ is a diagonal matrix, consisting of the weighting function in its diagonal elements, $\underline{t}$ is the target values of options prices, and $\underline{l}$ is the vector containing the unknown coefficients $\lambda_1, ..., \lambda_m$. The elements of $Y$ are given by $y_{ij} = \phi_j(\underline{U}_i)$, with $j = 1, ..., m$ and $i = n - 50, ..., n$.

## 5  SIMULATIONS

The data set for teh experiments consisted of call and put option contracts on the FTSE-100 Index, during the period February 1994 to December 1994. The date of maturity of all contracts was December 1994. Five pairs (Call and Put) of contracts at strike prices of 2925, 3025, 3125, 3225, and 3325.

The tracking behaviour of the EKF for one of the pairs is shown in Fig. 1 for a call/put pair with strike price 3125. Fig. 2 shows the trajectories of the underlying state vector for four different call/put option pairs. Table 2 shows the squared errors in the approximation errors computed over the last 100 days of data (allowing for an initial period of convergence of the recursive algorithms).

## 6  DISCUSSION

This paper presents a sequential approach to tracking the price of options contracts. The sequential approach is based on the Extended Kalman Filter algorithm, and I show how it may be used to identify a parametric model of the underlying nonlinear system. The model based approach of the finance community and the data driven approach of neural computing community lead to good estimates of the observed price of the options contracts when estimated in this manner.

In the state space formulation of the Black-Scholes model, the volatility and interest rate are estimated from the data. I trust the instantaneous picture presented by the model based approach, but reestimate the underlying parameters. This is different from conventional wisdom, where the risk free interest rate is set to some figure observed in the bond markets. The value of volatility that gives the correct options price through Black Scholes equation is called option implied volatility, and is usually different for different options. Option traders often use the differences in implied volatility to take trading positions. In the formulation presented here, there is an extra freedom coming in the form what one might call *implied interest rates*. It's difference from the interest rates observed in the markets might explain trader speculation about risk associated with a particular currency.

The derivatives of the RBF model output with respect to its inputs is easy to compute. Hutchinson *et al* use this to define a highly relevant performance measure

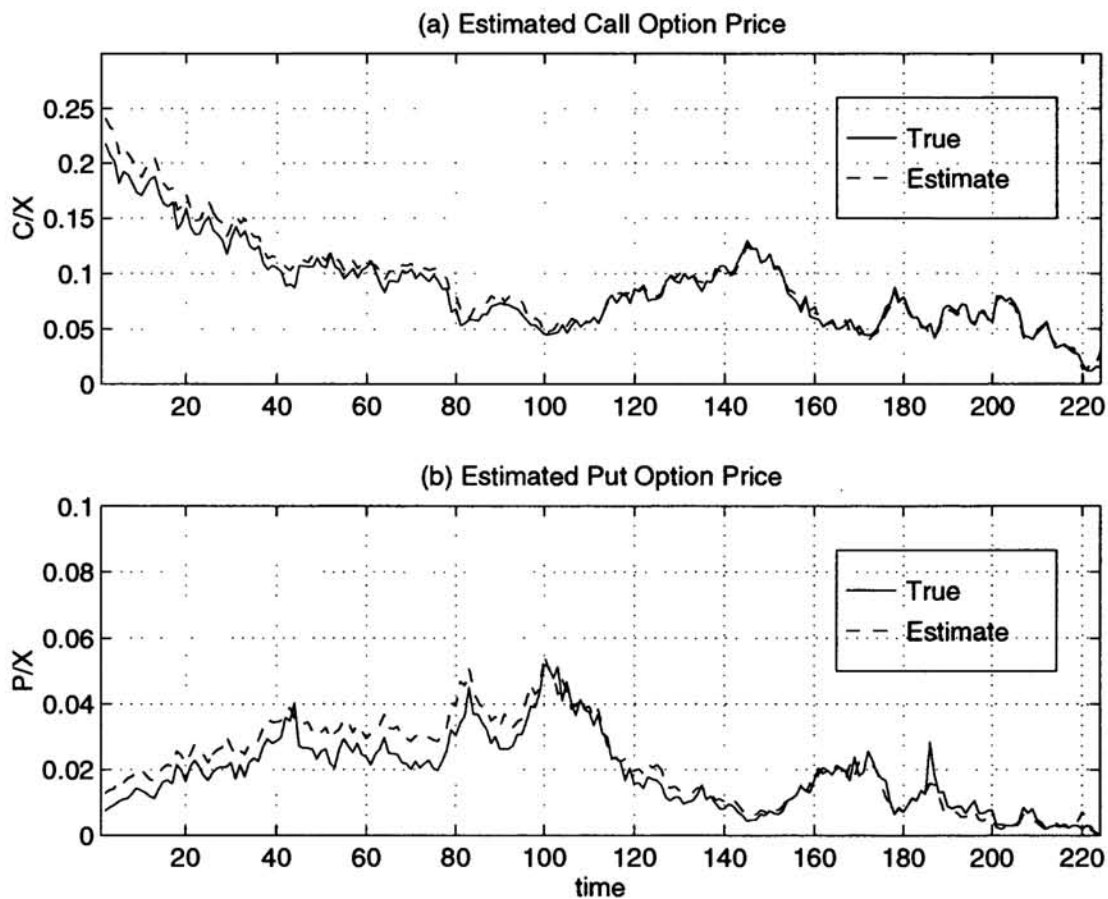

Figure 1: Tracking Black-Scholes Model with EKF; Estimates of Call and Put Prices

suitable to this particular application, namely the tracking error of a delta neutral portfolio. This is an evaluation that is somewhat unfair to the RBF model since at the time of training, the network is not shown the derivatives. An interesting combination of the work presented in this paper and Hutchinson *et al*'s performance measure is to train the neural network to approximate the observed option prices and simultaneously force the derivative network to approximate the delta observed in the markets.

# References

Bar-Shalom, Y. & Li, X-R. (1993), 'Estimation and Tracking: Principles, Techniques and Software', Artech House, London.

Candy, J. V. (1986), 'Signal Processing: The Model Based Aproach', McGraw-Hill, New York.

Hull, J. (1993), 'Options, Futures and Other Derivative Securities', Prentice Hall, NJ.

Hutchinson, J. M., Lo, A. W. & Poggio, T. (1994), 'A Nonparametric Approach to Pricing and Hedging Derivative Securities Via Learning Networks', *The Journal of Finance*, Vol XLIX, No. 3., 851-889.

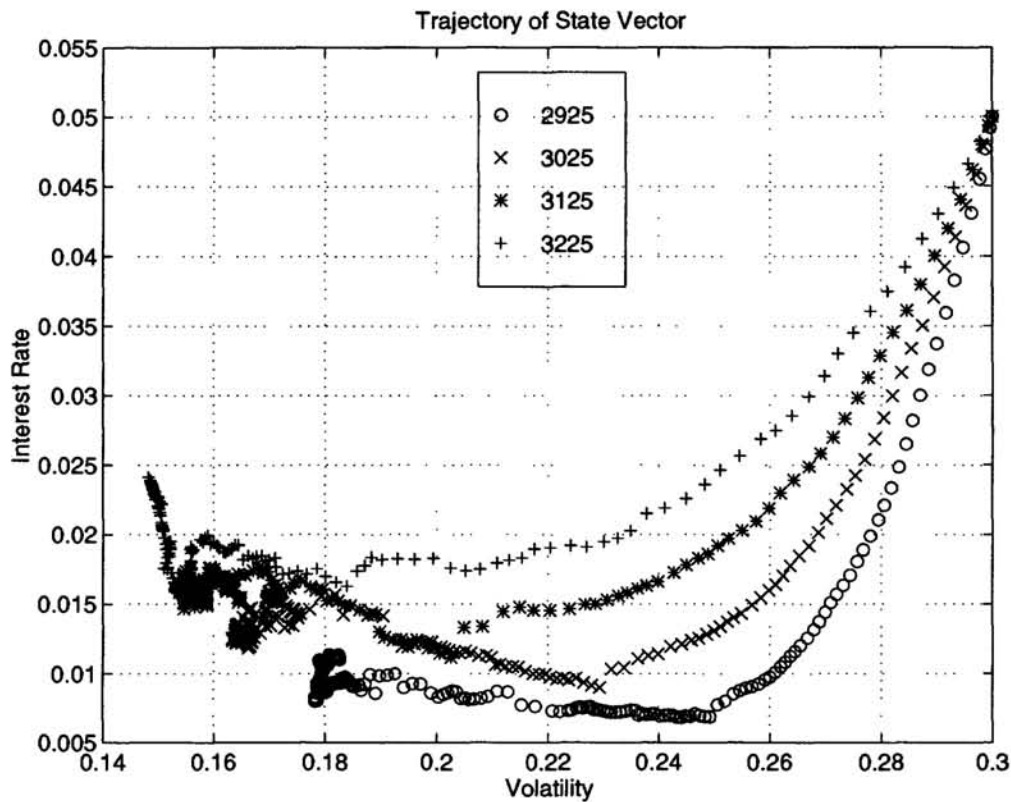

Figure 2: Tracking Black-Scholes Model with EKF; Estimates of Call and Put Prices and the Trajectory of the State Vector

Kadirkamanathan, V. & Niranjan, M (1993), 'A Function Estimation Approach to Sequential Learning with Neural Networks', *Neural Computation* **5**, pp. 954-975.

Lowe, D. (1995), 'On the use of Nonlocal and Non Positive Definite Basis Functions in Radial Basis Function Networks', Proceedings of the IEE Conference on Artificial Neural Networks, IEE Conference Publication No. 409, pp 206-211.

Niranjan, M., Cox, I. J., Hingorani, S. (1994), 'Recursive Estimation of Formants in Speech', Proceedings of the International Conference on Acoustics, Speech and Signal Processing, ICASSP '94, Adelaide.

Puskorius, G.V. & Feldkamp, L.A. (1994), 'Neurocontrol of Nonlinear Dynamical Systems with Kalman Filter-Trained Recurrent Networks', IEEE Transactions on Neural Networks, 5 (2), pp 279-297.
